# Extensions of a Theory of Networks for Approximation and Learning: Outliers and Negative Examples

Federico Girosi
AI Lab. M.I.T.
Cambridge, MA 02139

Tomaso Poggio
AI Lab. M.I.T.
Cambridge, MA 02139

Bruno Caprile
I.R.S.T.
Povo, Italy, 38050

## Abstract

Learning an input-output mapping from a set of examples can be regarded as synthesizing an approximation of a multi-dimensional function. From this point of view, this form of learning is closely related to regularization theory, and we have previously shown (Poggio and Girosi, 1990a, 1990b) the equivalence between regularization and a class of three-layer networks that we call regularization networks. In this note, we extend the theory by introducing ways of dealing with two aspects of learning: learning in presence of unreliable examples or outliers, and learning from positive *and* negative examples.

## 1 Introduction

In previous papers (Poggio and Girosi, 1990a, 1990b) we have shown the equivalence between certain regularization techniques and a class of three-layer networks – that we call regularization networks – which are related to the Radial Basis Functions interpolation method (Powell, 1987). In this note we indicate how it is possible to extend our theory of learning in order to deal with 1) occurence of unreliable examples, 2) negative examples. Both problems are also interesting from the point of view of classical approximation theory:

1. discounting "bad" examples corresponds to discarding, in the approximation of a function, data points that are outliers.

2. learning by using negative examples – in addition to positive ones – corresponds to approximating a function considering not only points which the function

ought to be close to, but also points – or regions – that the function must avoid.

## 2   Unreliable data

Suppose that a set $g = \{(\mathbf{x}_i, y_i) \in R^n \times R\}_{i=1}^N$ of data has been obtained by randomly sampling a function $f$, defined in $R^n$, in presence of noise, in a way that we can write

$$y_i = f(\mathbf{x}_i) + \epsilon_i, \quad i = 1, \ldots, N$$

where $\epsilon_i$ are independent random variables. We are interested in recovering an estimate of the function $f$ from the set of data $g$. Taking a probabilistic approach, we can regard the function $f$ as the realization of a random field with specified prior probability distribution. Consequently, the data $g$ and the function $f$ are non independent random variables, and, by using Bayes rule, it is possible to express the conditional probability $\mathcal{P}[f|g]$ of the function $f$, given the examples $g$, in terms of the prior probability of $f$, $\mathcal{P}[f]$, and the conditional probability of $g$ given $f$, $\mathcal{P}[g|f]$:

$$\mathcal{P}[f|g] \propto \mathcal{P}[g|f]\, \mathcal{P}[f]. \tag{1}$$

A common choice (Marroquin et al., 1987) for the prior probability distribution $\mathcal{P}[f]$ is

$$\mathcal{P}[f] \propto e^{-\lambda\|Pf\|^2} \tag{2}$$

where $P$ is a differential operator (the so called *stabilizer*), $\|\cdot\|$ is the $L^2$ norm, and $\lambda$ is a positive real number. This form of probability distribution assignes significant probability only to those functions for which the term $\|Pf\|^2$ is "small", that is to functions that do not vary too "quickly" in their domain.

If the noise is Gaussian, the probability $\mathcal{P}[g|f]$ can be written as:

$$\mathcal{P}[g|f] = \frac{2^N}{\pi^{\frac{N}{2}}} \prod_{i=1}^N \sqrt{\beta_i} e^{-\beta_i \epsilon_i^2} \tag{3}$$

where $\beta_i = \frac{1}{2\sigma_i^2}$, and $\sigma_i$ is the variance of the noise related to the $i$-th data point. The values of the variances are usually assumed to be equal to some known value $\sigma$, that reflects the accuracy of the measurement apparatus. However, in many cases we do not have access to such an information, and weaker assumptions have to be made. A fairly natural and general one consists in regarding the variances of the noise, as well as the function $f$, as random variables. Of course, some a priori knowledge about these variables, represented by an appropriate prior probability distribution, is needed. Let us denote by $\beta$ the set of random variables $\{\beta_i\}_{i=1}^N$. By

means of Bayes rule we can compute the joint probability of the variables $f$ and $\beta$. Assuming that the field $f$ and the set $\beta$ are conditionally independent we obtain:

$$\mathcal{P}[f, \beta | g] \propto \mathcal{P}[g | f, \beta] \, \mathcal{P}[f] \, \mathcal{P}[\beta] \tag{4}$$

where $\mathcal{P}[\beta]$ is the prior probability of the set of variances $\beta$ and $\mathcal{P}[g | f, \beta]$ is the same as in eq. (3). Given the posterior probability (4) we are mainly interested in computing an estimate of $f$. Thus what we really need to compute is the marginal posterior probability of $f$, $P_m[f]$, that is obtained integrating equation (4) over the variables $\beta_i$:

$$P_m[f] \propto \int_0^\infty \mathcal{P}[f, \beta | g] \prod_{i=1}^{N} d\beta_i \tag{5}$$

A simple way to obtain an estimate of the function $f$ from the probability distribution (5) consists in computing the so called MAP (Maximum A Posteriori) estimate, that is the function that maximizes the posterior probability $P_m[f]$. The problem of recovering the function $f$ from the set of data $g$, with partial information about the amount of Gaussian noise affecting the data, is therefore equivalent to solving an appropriate variational problem. The specific form of the functional that has to be maximized – or minimized – depends on the probability distributions $\mathcal{P}[f]$ and $\mathcal{P}[\beta]$.

Here we consider the following situation: we have knowledge that a given percentage, $(1 - \epsilon)$ of the data is characterized by a Gaussian noise distribution of variance $\sigma_1 = (2\beta_1)^{-\frac{1}{2}}$, whereas for the rest of the data the variance of the noise is a very large number $\sigma_2 = (2\beta_2)^{-\frac{1}{2}}$ (we will call these data "outliers"). This situation yields the following probability distribution:

$$\mathcal{P}[\beta] = \prod_{i=1}^{N} [(1 - \epsilon)\delta(\beta_i - \beta_1) + \epsilon \, \delta(\beta_i - \beta_2)] \, . \tag{6}$$

In this case, choosing $\mathcal{P}[f]$ as in eq. (2), we can show that $P_m[f] \propto e^{-H[f]}$, where

$$H[f] = \sum_{i=1}^{N} V(\epsilon_i) + \lambda \|Pf\|^2 \, . \tag{7}$$

Here $V$ represents the *effective potential*

$$V(x) = \beta_1 x^2 - \ln\left(1 + \frac{\epsilon}{1 - \epsilon}\sqrt{\frac{\beta_2}{\beta_1}}e^{x^2(\beta_1 - \beta_2)}\right) \, . \tag{8}$$

depicted in fig. (1) for different values of $\beta_2$.

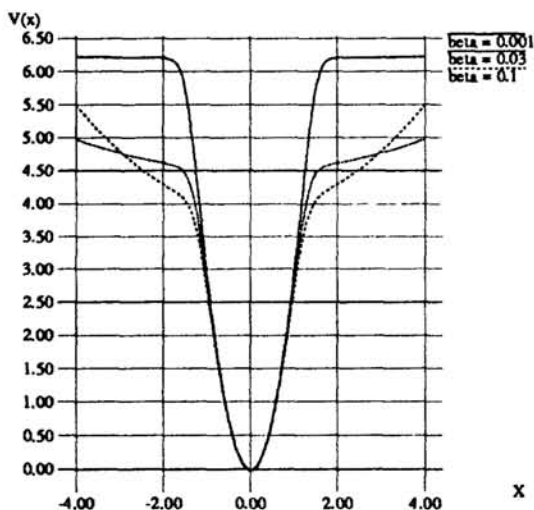

Figure 1: The effective potential $V(x)$ for $\epsilon = 0.1$, $\beta_1 = 3.0$ and three different values of $\beta_2$: 0.1, 0.03, 0.001

The MAP estimate is therefore obtained minimizing the functional (7). The first term enforces closeness to the data, while the second term enforces smoothness of the solution, the trade off between these two opposite tendencies being controlled by the parameter $\lambda$. Looking at fig. (1) we notice that, in the limit of $\beta_2 \rightarrow 0$, the effective potential $V$ is quadratic if the absolute value of its argument is smaller than a threshold, and constant otherwise (fig. 1). Therefore, data points are taken in account when the interpolation error is smaller than a threshold, and their contribution neglected otherwise.

If $\beta_1 = \beta_2 = \bar{\beta}$, that is if the distribution of the variables $\beta_i$ is a delta function centered on some value $\bar{\beta}$, the effective potential $V(x) = \bar{\beta}x^2$ is obtained. Therefore, this method becomes equivalent to the so called "regularization technique" (Tikhonov and Arsenin, 1977) that has been extensively used to solve *ill-posed problems*, of which the one we have just outlined is a particular example (Poggio and Girosi, 1990a, 1990b). Suitable choices of distribution $\mathcal{P}[\beta]$ result in other effective potentials (for example the potential $V(x) = \sqrt{\alpha^2 + x^2}$ can be obtained), and the corresponding estimators turn out to be similar to the well known *robust smoothing splines* (Eubank, 1988).

The functional (7), with the choice expressed by eq. (2), admits a simple physical interpretation. Let us consider for simplicity a function defined on a one-dimensional lattice. The value of the function $f(\mathbf{x}_i)$ at site $i$ is regarded as the position of a particle that can move only in the vertical direction. The particle is attracted – according to a spring-like potential $V$ – towards the data point and the neighboring

particles as well. The natural trend of the system will be to minimize its total energy which, in this scheme, is expressed by the functional (7): the first term is associated to the springs connecting the particle to the data point, and the second one, being associated to the the springs connecting neighboring particles, enforces the smoothness of the final configuration. Notice that the potential energy of the springs connecting the particle to the data point is not quadratic, as for the "standard" springs, resulting this in a non-linear relationship between the force and the elongation. The potential energy becomes constant when the elongation is larger than a fixed threshold, and the force (which is proportional to the first derivative of the potential energy) goes to zero. In this sense we can say that the springs "break" when we try to stretch them too much (Geiger and Girosi, 1990).

## 3   Negative examples

In many situations, further information about a function may consist in knowing that its value at some given point has to be far from a given value (which, in this context, can be considered as a "negative example"). We shall account for the presence of negative examples by adding to the functional (7) a quadratic repulsive term for each negative example (for a related trick, see Kass et al., 1987). However, the introduction of such a "repulsive spring" may make the functional (7) unbounded from below, because the repulsive terms tend to push the value of the function up to infinity. The simplest way to prevent this occurency is either to allow the spring constant to decrease with the increasing elongation, or, in the extreme case, to break at some point. Hence, we can use the same model of nonlinear spring of the previous section, and just reverse the sign of the associated potential. If $\{(\mathbf{t}_\alpha, y_\alpha) \in R^n \times R\}_{i=1}^K$ is the set of negative examples, and if we define $\Delta_\alpha = y_\alpha - f(\mathbf{t}_\alpha)$ the functional (7) becomes:

$$H[f] = \sum_{i=1}^N V(\Delta_i) - \sum_{\alpha=1}^K V(\Delta_\alpha) + \lambda \|Pf\|^2 \ .$$

## 4   Solution of the variational problem

An exhaustive discussion of the solution of the variational problem associated to the functional (7) cannot be given here. We refer the reader to the papers of Poggio and Girosi (1990a, 1990b) and Girosi, Poggio and Caprile (1990), and just sketch the form of the solution. In both cases of unreliable and negative data, it can be shown that the solution of the variational problem always has the form

$$f^*(\mathbf{x}) = \sum_{i=1}^N c_i G(\mathbf{x}; \mathbf{x}_i) + \sum_{i=1}^k \alpha_i \phi_i(\mathbf{x}) \tag{9}$$

where $G$ is the Green's function of the operator $\hat{P}P$ ($\hat{P}$ denoting the adjoint operator of $P$), and $\{\phi_i(\mathbf{x})\}_{i=1}^k$ is a basis of functions for the null space of $P$ (usually polynomials of low degree) and $\{c_i\}_{i=1}^N$ and $\{\alpha_i\}_{i=1}^k$ are coefficients to be computed.

Substituting the expansion (9) in the functional (7), the function $H^*(\mathbf{c}, \boldsymbol{\alpha}) = H[f^*]$ is defined. The vectors $\mathbf{c}$ and $\boldsymbol{\alpha}$ can then be found by minimizing the function $H^*(\mathbf{c}, \boldsymbol{\alpha})$.

We shall finally notice that the solution (9) has a simple interpretation in terms of feedforward networks with one layer of hidden units, of the same class of the regularization networks introduced in previous papers (Poggio and Girosi, 1990a, 1990b). The only difference between these networks and the regularization networks previously introduced consists in the function that has to be minimized in order to find the weights of the network.

## 5    Experimental Results

In this section we report two examples of the application of these techniques to very simple one-dimensional problems.

### 5.1    Unreliable data

The data set consisted of seven examples, randomly taken, within the interval $[-1, 1]$, from the graph of $f(x) = \cos(x)$. In order to create an outlier in the data set, the value of the fourth point has been substituted with the value 1.5. The Green's function of the problem was a Gaussian of variance $\sigma = 0.3$, the parameter $\epsilon$ was set to 0.1, the value of the regularization parameter $\lambda$ was $10^{-2}$, and the parameters $\beta_1$ and $\beta_2$ were set respectively to 10.0 and 0.003. With this choice of the parameters the effective potential was approximately constant for values of its argument larger than 1. In figure (2a) we show the result that is obtained after only 10 iterations of gradient descent: the spring of the outlier breaks, and it does not influence the solution any more. The "hole" that the solution shows nearby the outlier is a combined effect of the fact that the variance of the Gaussian Green's function is small ($\sigma = 0.3$), and of the lack of data next to the outlier itself.

### 5.2    Negative examples

Again data to be approximated came from a random sampling of the function $f(x) = \cos(x)$, in the interval $[-1, 1]$. The fourth data point was selected as the negative example, and the parameters were set in a way that its spring would break when the elongation exceeded the value 1. In figure (2b) we show a result obtained with 500 iterations of a stochastic gradient descent algorithm, with a Gaussian Green's function of variance $\sigma = 0.4$.

*Acknowledgements* We thank Cesare Furlanello for useful discussions and for a critical reading of the manuscript.

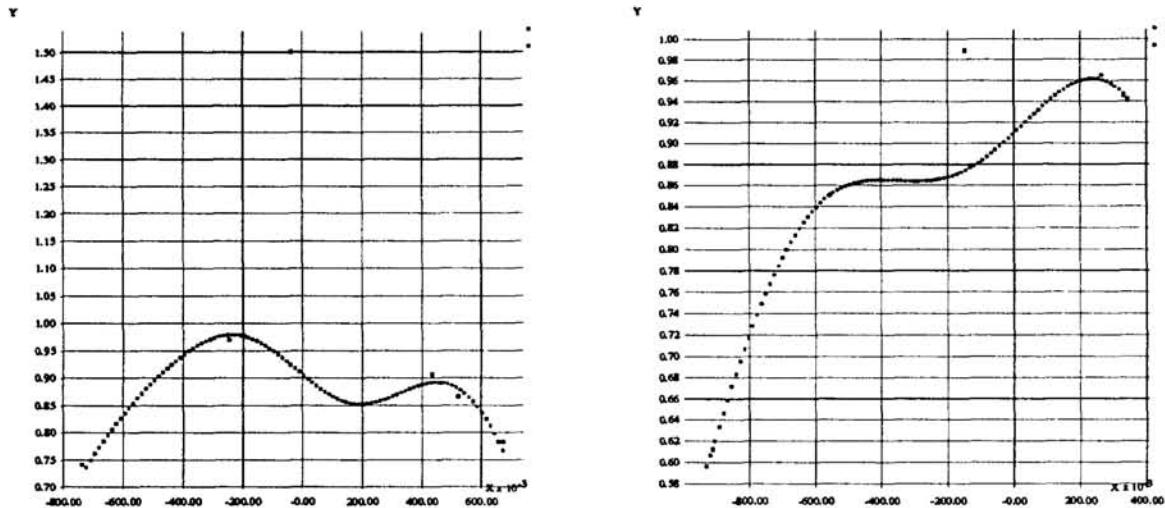

Figure 2: (a) Approximation in presence of an outlier (the data point whose value is 1.5). (b) Approximation in presence of a negative example.

# References

[1] R.L. Eubank. *Spline Smoothing and Nonparametric Regression*, volume 90 of *Statistics: Textbooks and Monographs*. Marcel Dekker, Inc., New York, 1988.

[2] D. Geiger and F. Girosi. Parallel and deterministic algorithms for MRFs: surface reconstruction and integration. In O. Faugeras, editor, *Lecture Notes in Computer Science, Vol. 427: Computer Vision – ECCV 90*. Springer-Verlag, Berlin, 1990.

[3] F. Girosi, T. Poggio, and B. Caprile. Extensions of a theory of networks for approximation and learning: outliers and negative examples. A.I. Memo 1220, Artificial Intelligence Laboratory, Massachusetts Institute of Technology, 1990.

[4] M. Kass, A. Witkin, and D. Terzopoulos. Snakes: Active contour models. In *Proceedings of the First International Conference on Computer Vision*, London, 1987. IEEE Computer Society Press, Washington, D.C.

[5] J. L. Marroquin, S. Mitter, and T. Poggio. Probabilistic solution of ill-posed problems in computational vision. *J. Amer. Stat. Assoc.*, 82:76–89, 1987.

[6] T. Poggio and F. Girosi. A theory of networks for learning. *Science*, 247:978–982, 1990a.

[7] T. Poggio and F. Girosi. Networks for approximation and learning. *Proceedings of the IEEE*, 78(9), September 1990b.

[8] M. J. D. Powell. Radial basis functions for multivariable interpolation: a review. In J. C. Mason and M. G. Cox, editors, *Algorithms for Approximation*. Clarendon Press, Oxford, 1987.

[9] A. N. Tikhonov and V. Y. Arsenin. *Solutions of Ill-posed Problems*. W. H. Winston, Washington, D.C., 1977.
